# Transition Point Dynamic Programming

**Kenneth M. Buckland***
Dept. of Electrical Engineering
University of British Columbia
Vancouver, B.C, Canada V6T 1Z4
buckland@pmc-sierra.bc.ca

**Peter D. Lawrence**
Dept. of Electrical Engineering
University of British Columbia
Vancouver, B.C, Canada V6T 1Z4
peterl@ee.ubc.ca

## Abstract

Transition point dynamic programming (TPDP) is a memory-based, reinforcement learning, direct dynamic programming approach to adaptive optimal control that can reduce the learning time and memory usage required for the control of continuous stochastic dynamic systems. TPDP does so by determining an ideal set of transition points (TPs) which specify only the control action *changes* necessary for optimal control. TPDP converges to an ideal TP set by using a variation of Q-learning to assess the merits of adding, swapping and removing TPs from states throughout the state space. When applied to a race track problem, TPDP learned the optimal control policy much sooner than conventional Q-learning, and was able to do so using less memory.

## 1  INTRODUCTION

Dynamic programming (DP) approaches can be utilized to determine optimal control policies for continuous stochastic dynamic systems when the state spaces of those systems have been quantized with a resolution suitable for control (Barto *et al.*, 1991). DP controllers, in their simplest form, are memory-based controllers that operate by repeatedly updating cost values associated with every state in the discretized state space (Barto *et al.*, 1991). In a state space of any size the required quantization can lead to an excessive memory requirement, and a related increase in learning time (Moore, 1991). This is the "curse of dimensionality".

Q-learning (Watkins, 1989, Watkins *et al.*, 1992) is a direct form of DP that avoids explicit system modeling – thereby reducing the memory required for DP control. Further reductions are possible if Q-learning is modified so that its DP cost values (Q-values) are associated only with states where control action *changes* need to be specified. Transition point dynamic programming (TPDP), the control approach described in this paper, is designed to take advantage of this DP memory reduction possibility by determining the states where control action changes must be specified for optimal control, and what those optimal changes are.

## 2    GENERAL DESCRIPTION OF TPDP

### 2.1    TAKING ADVANTAGE OF INERTIA

TPDP is suited to the control of continuous stochastic dynamic systems that have inertia. In such systems "uniform regions" are likely to exist in the state space where all of the (discretized) states have the same optimal control action (or the same set of optimal actions[1]). Considering one such uniform region, if the optimal action for that region is specified at the "boundary states" of the region and then *maintained* throughout the region until it is left and another uniform region is entered (where another set of boundary states specify the next action), none of the "dormant states" in the middle of the region need to specify any actions themselves. Thus dormant states do not have to be represented in memory. This is the basic premise of TPDP.

The association of optimal actions with boundary states is done by "transition points" (TPs) at those states. Boundary states include all of the states that can be reached from outside a uniform region when that region is entered as a result of stochastic state transitions. The boundary states of any one uniform region form a hyper-surface of variable thickness which may or may not be closed. The TPs at boundary states must be represented in memory, but if they are small in number compared to the dormant states the memory savings can be significant.

### 2.2    ILLUSTRATING THE TPDP CONCEPT

Figure 1 illustrates the TPDP concept when movement control of a "car" on a one dimensional track is desired. The car, with some initial positive velocity to the right, must pass Position A and return to the left. The TPs in Figure 1 (represented by boxes) are located at boundary states. The shaded regions indicate all of the states that the system can possibly move through given the actions specified at the boundary states and the stochastic response of the car. Shaded states without TPs are therefore dormant states. Uniform regions consist of adjacent boundary states where the same action is specified, as well as the shaded region through which that action is maintained before another boundary is encountered. Boundary states that do not seem to be on the main state transition routes (the one identified in Figure 1 for example) ensure that any stochastic deviations from those routes are realigned. Unshaded states are "external states" the system does not reach.

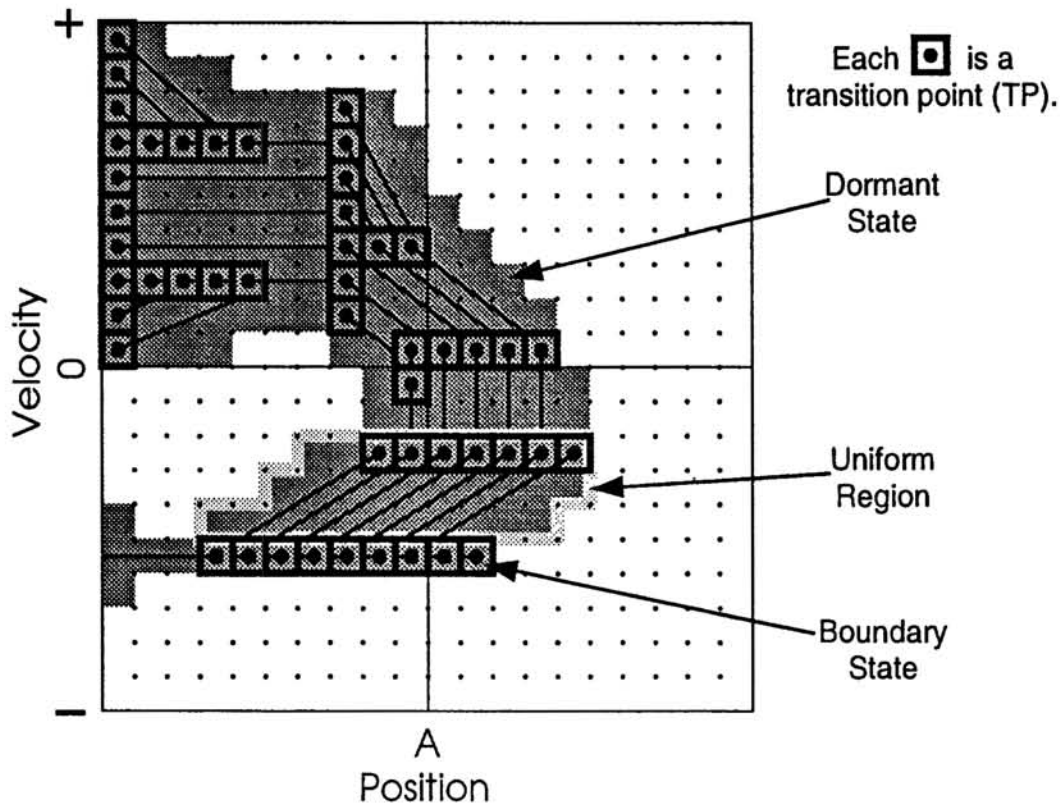

Figure 1: Application of TPDP to a One Dimension Movement Control Task

## 2.3   MINIMAL TP OPTIMAL CONTROL

The main benefit of the TPDP approach is that, where uniform regions exist, they can be represented by a relatively small number of DP elements (TPs) – depending on the shape of the boundaries and the size of the uniform regions they encompass. This reduction in memory usage results in an accompanying reduction in the learning time required to learn optimal control policies (Chapman *et al.*, 1991).

TPDP operates by learning optimal points of transition in the control action specification, where those points can be accurately located in highly resolved state spaces. To do this TPDP must determine which states are boundary states that should have TPs, and what actions those TPs should specify. In other words, TPDP must find the right TPs for the right states. When it has done so, "minimal TP optimal control" has been achieved. That is, optimal control with a minimal set of TPs.

## 3   ACHIEVING MINIMAL TP OPTIMAL CONTROL

### 3.1   MODIFYING A SET OF TPs

Given an arbitrary initial set of TPs, TPDP must modify that set so that it is transformed into a minimal TP optimal control set. Modifications can include the "addition" and "removal" of TPs throughout the state space, and the "swapping" of one TP for another (each specifying a different action) at the same state. These

modifications are performed one at a time in arbitrary order, and can continue indefinitely. TPDP operates so that each TP modification results in an incremental movement towards minimal TP optimal control (Buckland, 1994).

## 3.2   Q-LEARNING

TPDP makes use of Q-learning (Watkins, 1989, Watkins *et al.*, 1992) to modify the TP set. Normally Q-learning is used to determine the optimal control policy $\mu$ for a stochastic dynamic system subjected to immediate costs $c(i, u)$ when action $u$ is applied in each state $i$ (Barto *et al.*, 1991). Q-learning makes use of "Q-values" $Q(i, u)$, which indicate the expected total infinite-horizon discounted cost if action $u$ is applied in state $i$, and actions defined by the existing policy $\mu$ are applied in all future states. Q-values are learned by using the following updating equation:

$$Q_{t+1}(s_t, u_t) = (1 - \alpha_t)Q_t(s_t, u_t) + \alpha_t \left[ c(s_t, u_t) + \gamma V_t(s_{t+1}) \right] \qquad (1)$$

Where $\alpha_t$ is the update rate, $\gamma$ is the discount factor, and $s_t$ and $u_t$ are respectively the state at time step $t$ and the action taken at that time step (all other Q-values remain the same at time step $t$). The evaluation function value $V_t(i)$ is set to the lowest Q-value action of all those possible $U(i)$ in each state $i$:

$$V_t(i) = \min_{u \in U(i)} Q_t(i, u) \qquad (2)$$

If Equations 1 and 2 are employed during exploratory movement of the system, it has been proven that convergence to optimal Q-values $Q^*(i, u)$ and optimal evaluation function values $V_{\mu^*}(i)$ will result (given that the proper constraints are followed, Watkins, 1989, Watkins *et al.*, 1992, Jaakkola *et al.*, 1994). From these values the optimal action in each state can be determined (the action that fulfills Equation 2).

## 3.3   ASSESSING TPs WITH Q-LEARNING

TPDP uses Q-learning to determine how an existing set of TPs should be modified to achieve minimal TP optimal control. Q-values can be associated with TPs, and the Q-values of two TPs at the same "TP state", each specifying different actions, can be compared to determine which should be maintained at that state – that is, which has the lower Q-value. This is how TPs are swapped (Buckland, 1994).

States which do not have TPs, "non-TP states", have no Q-values from which evaluation function values $V_t(i)$ can be determined (using Equation 2). As a result, to learn TP Q-values, Equation 1 must be modified to facilitate Q-value updating when the system makes $d$ state transitions from one TP state through a number of non-TP states to another TP state:

$$Q_{t+d}(s_t, u_t) = (1 - \alpha_t)Q_t(s_t, u_t) + \alpha_t \left[ \left( \sum_{n=0}^{d-1} \gamma^n c(s_{t+n}, u_t) \right) + \gamma^d V_t(s_{t+d}) \right] \qquad (3)$$

When $d = 1$, Equation 3 takes the form of Equation 1. When $d > 1$, the intervening non-TP states are effectively ignored and treated as inherent parts of the stochastic dynamic behavior of the system (Buckland, 1994).

If Equation 3 is used to determine the costs incurred when *no* action is specified at a state (when the action specified at some previous state is maintained), an "R-value" $R(i)$ is the result. R-values can be used to expediently add and remove TPs

from each state. If the Q-value of a TP is less than the R-value of the state it is associated with, then it is worthwhile having that TP at that state; otherwise it is not (Buckland, 1994).

## 3.4   CONVERGENCE TO MINIMAL TP OPTIMAL CONTROL

It has been proven that a random sequence of TP additions, swaps and removals attempted at states throughout the state space will result in convergence to minimal TP optimal control (Buckland, 1994). This proof depends mainly on all TP modifications "locking-in" any potential cost reductions which are discovered as the result of learning exploration.

The problem with this proof of convergence, and the theoretical form of TPDP described up to this point, is that each modification to the existing set of TPs (each addition, swap and removal) requires the determination of Q-values and R-values which are negligibly close to being exact. This means that a complete session of Q-learning must occur for every TP modification.[2] The result is excessive learning times – a problem circumvented by the practical form of TPDP described next.

# 4   PRACTICAL TPDP

## 4.1   CONCURRENT TP ASSESSMENT

To solve the problem of the protracted learning time required by the theoretical form of TPDP, many TP modifications can be assessed concurrently. That is, Q-learning can be employed not just to determine the Q-values and R-values for a single TP modification, but instead to learn these values for a number of concurrent modifications. Further, the modification attempts, and the learning of the values required for them, need not be initiated simultaneously. The determination of each value can be made part of the Q-learning process whenever new modifications are randomly attempted. This approach is called "Practical TPDP". Practical TPDP consists of a continually running Q-learning process (based on Equations 2 and 3), where the Q-values and R-values of a constantly changing set of TPs are learned.

## 4.2   USING WEIGHTS FOR CONCURRENT TP ASSESSMENT

The main difficulty that arises when TPs are assessed concurrently is that of determining when an assessment is complete. That is, when the Q-values and R-values associated with each TP have been learned well enough for a TP modification to be made based on them. The technique employed to address this problem is to associate a "weight" $w(i, u)$ with each TP that indicates the general merit of that TP. The basic idea of weights is to facilitate the random *addition* of trial TPs to a TP "assessment group" with a low initial weight $w_{initial}$. The Q-values and R-values of the TPs in the assessment group are learned in an ongoing Q-learning process, and the weights of the TPs are adjusted heuristically using those values. Of those TPs at any state $i$ whose weights $w(i, u)$ have been increased above $w_{thr}$

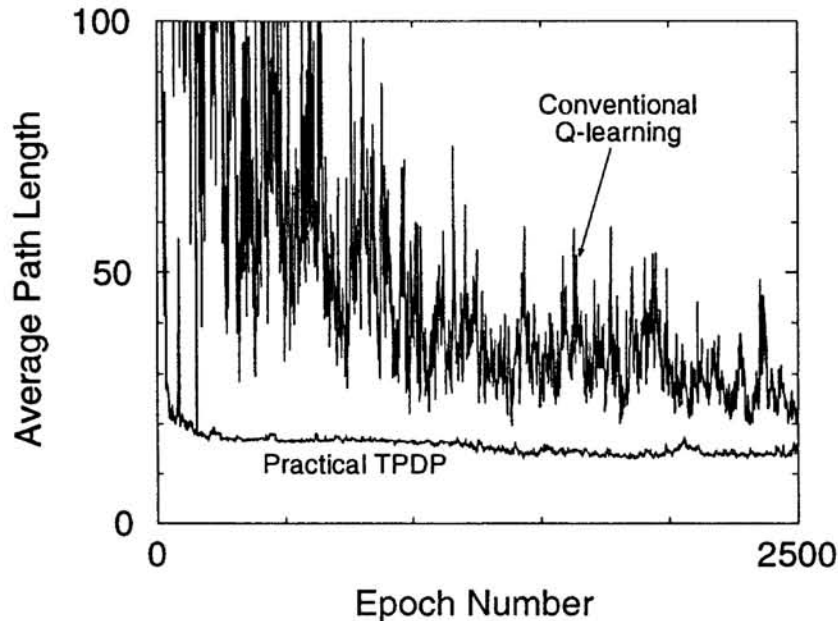

Figure 2: Performance of Practical TPDP on a Race Track Problem

($w_{\text{initial}} < w_{\text{thr}} < w_{\text{max}}$), the one with the lowest Q-value $Q(i, u)$ is *swapped* into the "policy TP" role for that state. The heuristic weight adjustment rules are:

1. New, trial TPs are given an initial weight of $w_{\text{initial}}$ ($0 < w_{\text{initial}} < w_{\text{thr}}$).

2. Each time the Q-value of a TP is updated, the weight $w(i, u)$ of that TP is incremented if $Q(i, u) < R(i)$ and decremented otherwise.

3. Each TP weight $w(i, u)$ is limited to a maximum value of $w_{\text{max}}$. This prevents any one weight from becoming so large that it cannot readily be reduced again.

4. If a TP weight $w(i, u)$ is decremented to 0 the TP is *removed*.

An algorithm for Practical TPDP implementation is described in Buckland (1994).

## 4.3   PERFORMANCE OF PRACTICAL TPDP

Practical TPDP was applied to a continuous version of a control task described by Barto *et al.* (1991) – that of controlling the acceleration of a car down a race track (specifically the track shown in Figures 3 and 4) when that car randomly experiences control action non-responsiveness. As shown in Figure 2 (each epoch in this Figure consisted of 20 training trials and 500 testing trials), Practical TPDP learned the optimal control policy much sooner than conventional Q-learning, and it was able to do so when limited to only 15% of the possible number of TPs (Buckland, 1994). The possible number of TPs is the full set of Q-values required by conventional Q-learning (one for each possible state and action combination).

The main advantage of Practical TPDP is that it facilitates rapid learning of preliminary control policies. Figure 3 shows typical routes followed by the car early

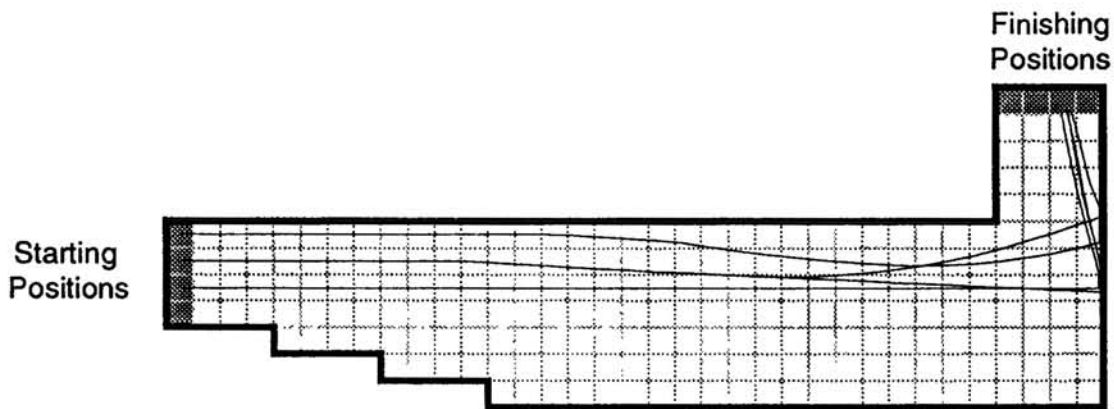

Figure 3: Typical Race Track Routes After 300 Epochs

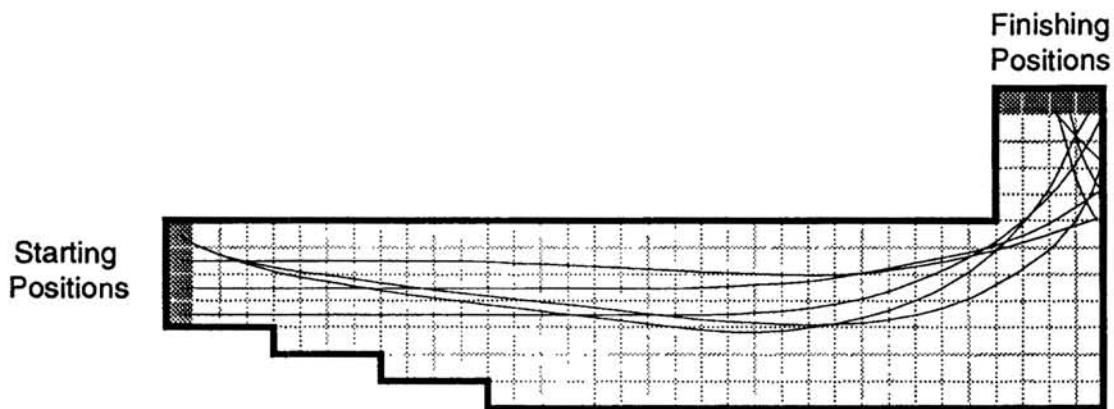

Figure 4: Typical Race Track Routes After 1300 Epochs

in the learning process. With the addition of relatively few TPs, the policy of accelerating wildly down the track, smashing into the wall and continuing on to the finishing positions was learned. Further learning centered around this preliminary policy led to the optimal policy of sweeping around the left turn. Figure 4 shows typical routes followed by the car during this shift in the learned policy – a shift indicated by a slight drop in the learning curve shown in Figure 2 (around 1300 epochs). After this shift, learning progressed rapidly until roughly optimal policies were consistently followed.

A problem which occurs in Practical TPDP is that of the addition of superfluous TPs after the optimal policy has basically been learned. The reasons this occurs are described in Buckland (1994), as well as a number of solutions to the problem.

## 5   CONCLUSION

The practical form of TPDP performs very well when compared to conventional Q-learning. When applied to a race track problem it was able to learn optimal policies more quickly while using less memory. Like Q-learning, TPDP has all the

advantages and disadvantages that result from it being a direct control approach that develops no explicit system model (Watkins, 1989, Buckland, 1994).

In order to take advantage of the sparse memory usage that occurs in TPDP, TPs are best represented by ACAMs (associative content addressable memories, Atkeson, 1989). A localized neural network design which operates as an ACAM and which facilitates Practical TPDP control is described in Buckland *et al.* (1993) and Buckland (1994).

The main idea of TPDP is to, "try this for a while and see what happens". This is a potentially powerful approach, and the use of TPs associated with abstracted control actions could be found to have substantial utility in hierarchical control systems.

## Acknowledgements

Thanks to John Ip for his help on this work. This work was supported by an NSERC Postgraduate Scholarship, and NSERC Operating Grant A4922.

## Footnotes

*Now at: PMC-Sierra Inc., 8501 Commerce Court, Burnaby, B.C., Canada V5A 4N3.

[1]The simplifying assumption that there is only one optimal action in each uniform region will be made throughout this paper. TPDP operates the same regardless.

[2]The TPDP proof allows for more than one TP swap to be assessed simultaneously, but this does little to reduce the overall problem being described (Buckland, 1994).

## References

Atkeson, C. G. (1989), "Learning arm kinematics and dynamics", *Annual Review of Neuroscience*, vol. 12, 1989, pp. 157-183.

Barto, A. G., S. J. Bradtke and S. P. Singh (1991), "Real-time learning and control using asynchronous dynamic programming", *COINS Technical Report 91-57*, University of Massachusetts, Aug. 1991.

Buckland, K. M. and P. D. Lawrence (1993), "A connectionist approach to direct dynamic programming control", *Proc. of the IEEE Pacific Rim Conf. on Communications, Computers and Signal Processing*, Victoria, 1993, vol. 1, pp. 284-287.

Buckland, K. M. (1994), *Optimal Control of Dynamic Systems Through the Reinforcement Learning of Transition Points*, Ph.D. Thesis, Dept. of Electrical Engineering, University of British Columbia, 1994.

Chapman, D. and L. P. Kaelbling (1991), "Input generalization in delayed reinforcement-learning: an algorithm and performance comparisons", *Proc. of the 12th Int. Joint Conf. on Artificial Intelligence*, Sydney, Aug. 1991, pp. 726-731.

Jaakkola, T., M. I. Jordan and S. P. Singh (1994), "Stochastic convergence of iterative DP algorithms", *Advances in Neural Information Processing Systems 6*, eds.: J. D. Cowen, G. Tesauro and J. Alspector, San Francisco, CA: Morgan Kaufmann Publishers, 1994.

Moore, A. W. (1991), "Variable resolution dynamic programming: efficiently learning action maps in multivariate real-valued state-spaces", *Machine Learning: Proc. of the 8th Int. Workshop*, San Mateo, CA: Morgan Kaufmann Publishers, 1991.

Watkins, C. J. C. H. (1989), *Learning from Delayed Rewards*, Ph.D. Thesis, Cambridge University, Cambridge, England, 1989.

Watkins, C. J. C. H. and P. Dayan (1992), "Q-learning", *Machine Learning*, vol. 8, 1992, pp. 279-292.